# Applying Metric-Trees to Belief-Point POMDPs

**Joelle Pineau, Geoffrey Gordon**
School of Computer Science
Carnegie Mellon University
Pittsburgh, PA 15213
{jpineau,ggordon}@cs.cmu.edu

**Sebastian Thrun**
Computer Science Department
Stanford University
Stanford, CA 94305
thrun@stanford.edu

## Abstract

Recent developments in grid-based and point-based approximation algorithms for POMDPs have greatly improved the tractability of POMDP planning. These approaches operate on sets of belief points by individually learning a value function for each point. In reality, belief points exist in a highly-structured metric simplex, but current POMDP algorithms do not exploit this property. This paper presents a new metric-tree algorithm which can be used in the context of POMDP planning to sort belief points spatially, and then perform fast value function updates over groups of points. We present results showing that this approach can reduce computation in point-based POMDP algorithms for a wide range of problems.

## 1  Introduction

Planning under uncertainty is a central problem in the field of robotics as well as many other AI applications. In terms of representational effectiveness, the Partially Observable Markov Decision Process (POMDP) is among the most promising frameworks for this problem. However the practical use of POMDPs has been severely limited by the computational requirement of planning in such a rich representation. POMDP planning is difficult because it involves learning action selection strategies contingent on all possible types of state uncertainty. This means that whenever the robot's world state cannot be observed, the planner must maintain a *belief* (namely a probability distribution over possible states) to summarize the robot's recent history of actions taken and observations received. The POMDP planner then learns an optimal future action selection for each possible belief. As the planning horizon grows (linearly), so does the number of possible beliefs (exponentially), which causes the computational intractability of exact POMDP planning.

In recent years, a number of approximate algorithms have been proposed which overcome this issue by simply refusing to consider all possible beliefs, and instead selecting (and planning for) a small set of representative belief points. During execution, should the robot encounter a belief for which it has no plan, it finds the nearest known belief point and follows its plan. Such approaches, often known as grid-based [1, 4, 13], or point-based [8, 9] algorithms, have had significant success with increasingly large planning domains. They formulate the plan optimization problem as a value iteration procedure, and estimate the cost/reward of applying a sequence of actions from a given belief point. The value of

each action sequence can be expressed as an $\alpha$-*vector*, and a key step in many algorithms consists of evaluating many candidate $\alpha$-vectors (set $\Gamma$) at each belief point (set $B$).

These $B \times \Gamma$ (point-to-vector) comparisons—which are typically the main bottleneck in scaling point-based algorithms—are reminiscent of many $M \times N$ comparison problems that arise in statistical learning tasks, such as kNN, mixture models, kernel regression, etc. Recent work has shown that for these problems, one can significantly reduce the number of necessary comparisons by using appropriate metric data structures, such as KD-trees and ball-trees [3, 6, 12]. Given this insight, we extend the metric-tree approach to POMDP planning, with the specific goal of reducing the number of $B \times \Gamma$ comparisons. This paper describes our algorithm for building and searching a metric-tree over belief points.

In addition to improving the scalability of POMDP planning, this approach features a number of interesting ideas for generalizing metric-tree algorithms. For example, when using trees for POMDPs, we move away from point-to-point search procedures for which the trees are typically used, and leverage metric constraints to prune point-to-vector comparisons. We show how it is often possible to evaluate the usefulness of an $\alpha$-vector over an entire sub-region of the belief simplex without explicitly evaluating it at each belief point in that sub-region. While our new metric-tree approach offers significant potential for all point-based approaches, in this paper we apply it in the context of the PBVI algorithm [8], and show that it can effectively reduce computation without compromising plan quality.

## 2    Partially Observable Markov Decision Processes

We adopt the standard POMDP formulation [5], defining a problem by the $n$-tuple: $\{S, A, Z, T, O, R, \gamma, b_0\}$, where $S$ is a set of (discrete) world states describing the problem domain, $A$ is a set of possible actions, and $Z$ is a set of possible observations providing (possibly noisy and/or partial) state information. The distribution $T(s, a, s')$ describes state-to-state transition probabilities; distribution $O(s, a, z)$ describes observation emission probabilities; function $R(s, a)$ represents the reward received for applying action $a$ in state $s$; $\gamma$ represents the discount factor; and $b_0$ specifies the initial belief distribution. An $|S|$-dimensional vector, $b_t$, represents the agent's belief about the state of the world at time $t$, and is expressed as a probability distribution over states. This belief is updated after each time step—to reflect the latest pair $(a_{t-1}, z_t)$—using a Bayesian filter: $b_t(s') := c\, O(s', a_{t-1}, z_t) \sum_{s \in S} T(s, a_{t-1}, s') b_{t-1}(s)$, where $c$ is a normalizing constant.

The goal of POMDP planning is to find a sequence of actions maximizing the expected sum of rewards $E[\sum_t \gamma^t R(s_t, a_t)]$, for all belief. The corresponding value function can be formulated as a Bellman equation: $V(b) = \max_{a \in A} \left[ R(b, a) + \gamma \sum_{b' \in B} T(b, a, b') V(b') \right]$

By definition there exist an infinite number of belief points. However when optimized exactly, the value function is always piecewise linear and convex in the belief (Fig. 1a). After $n$ value iterations, the solution consists of a finite set of $\alpha$-vectors: $V_n = \{\alpha_0, \alpha_1, ..., \alpha_m\}$. Each $\alpha$-vector represents an $|S|$-dimensional hyper-plane, and defines the value function over a bounded region of the belief: $V_n(b) = \max_{\alpha \in V_n} \sum_{s \in S} \alpha(s) b(s)$. When performing exact value updates, the set of $\alpha$-vectors can (and often does) grow exponentially with the planning horizon. Therefore exact algorithms tend to be impractical for all but the smallest problems. We leave out a full discussion of exact POMDP planning (see [5] for more) and focus instead on the much more tractable point-based approximate algorithm.

## 3    Point-based value iteration for POMDPs

The main motivation behind the point-based algorithm is to exploit the fact that most beliefs are never, or very rarely, encountered, and thus resources are better spent planning

for those beliefs that are most likely to be reached. Many classical POMDP algorithms do not exploit this insight. Point-based value iteration algorithms on the other hand apply value backups only to a finite set of pre-selected (and likely to be encountered) belief points $B = \{b_0, b_1, ..., b_q\}$. They initialize a separate $\alpha$-vector for each selected point, and repeatedly update the value of that $\alpha$-vector. As shown in Figure 1b, by maintaining a full $\alpha$-vector for each belief point, we can preserve the piecewise linearity and convexity of the value function, and define a value function over the entire belief simplex. This is an approximation, as some vectors may be missed, but by appropriately selecting points, we can bound the approximation error (see [8] for details).

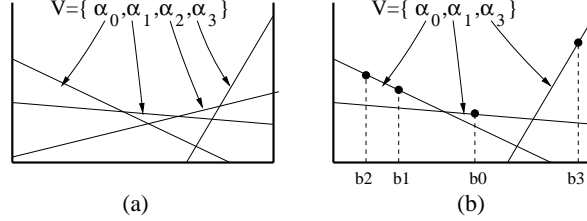

Figure 1: (a) Value iteration with exact updates. (b) Value iteration with point-based updates.

There are generally two phases to point-based algorithms. First, a set of belief points is selected, and second, a series of backup operations are applied over $\alpha$-vectors for that set of points. In practice, steps of value iteration and steps of belief set expansion can be repeatedly interleaved to produce an anytime algorithm that can gradually trade-off computation time and solution quality. The question of how to best select belief points is somewhat orthogonal to the ideas in this paper and is discussed in detail in [8]. We therefore focus on describing how to do point-based value backups, before showing how this step can be significantly accelerated by the use of appropriate metric data structures.

The traditional value iteration POMDP backup operation is formulated as a dynamic program, where we build the $n$-th horizon value function $V$ from the previous solution $V'$:

$$V(b) = \max_{a \in A} \left[ \sum_{s \in S} R(s,a)b(s) + \gamma \sum_{z \in Z} \max_{\alpha' \in V'} \sum_{s \in S} \sum_{s' \in S} T(s,a,s')O(z,s',a)\alpha'(s')b(s) \right] \quad (1)$$

$$= \max_{a \in A} \left[ \sum_{z \in Z} \max_{\alpha' \in V'} \left[ \sum_{s \in S} \frac{R(s,a)}{|Z|} b(s) + \gamma \sum_{s \in S} \sum_{s' \in S} T(s,a,s')O(z,s',a)\alpha'(s')b(s) \right] \right]$$

To plan for a finite set of belief points $B$, we can modify this operation such that only one $\alpha$-vector per belief point is maintained and therefore we only consider $V(b)$ at points $b \in B$. This is implemented using three steps. First, we take each vector in $V'$ and project it backward (according to the model) for a given action, observation pair. In doing so, we generate intermediate sets $\Gamma^{a,z}, \forall a \in A, \forall z \in Z$:

$$\Gamma^{a,z} \leftarrow \alpha_i^{a,z}(s) = \frac{R(s,a)}{|Z|} + \gamma \sum_{s' \in S} T(s,a,s')O(z,s',a)\alpha_i'(s'), \forall \alpha_i' \in V' \text{ (Step 1)} \quad (2)$$

Second for each $b \in B$, we construct $\Gamma^a$ ($\forall a \in A$). This sum over observations[1] includes the maximum $\alpha^{a,z}$ (at a given $b$) from each $\Gamma^{a,z}$:

$$\Gamma_b^a = \sum_{z \in Z} \underset{\alpha \in \Gamma^{a,z}}{\operatorname{argmax}} (\alpha \cdot b) \text{ (Step 2)} \quad (3)$$

Finally, we find the best action for each belief point:

$$V \quad \leftarrow \quad \underset{\Gamma_b^a, \forall a \in A}{\operatorname{argmax}}(\Gamma_b^a \cdot b), \quad \forall b \in B \text{ (Step 3)} \tag{4}$$

The main bottleneck in applying point-based algorithms to larger POMDPs is in step 2 where we perform a $B \times \Gamma$ comparison[2]: for every $b \in B$, we must find the best vector from a given set $\Gamma^{a,z}$. This is usually implemented as a sequential search, exhaustively comparing $\alpha \cdot b$ for every $b \in B$ and every $\alpha \in \Gamma^{a,z}$, in order to find the best $\alpha$ at each $b$ (with overall time-complexity $O(|A| |Z| |S| |B| |V'|)$). While this is not entirely unreasonable, it is by far the slowest step. It also completely ignores the highly structured nature of the belief space.

Belief points exist in a metric space and there is much to be gained from exploiting this property. For example, given the piecewise linearity and convexity of the value function, it is more likely that two nearby points will share similar values (and policies) than points that are far away. Consequently it could be much more efficient to evaluate an $\alpha$-vector over sets of nearby points, rather than by exhaustively looking at all the points separately. In the next section, we describe a new type of metric-tree which structures data points based on a distance metric over the belief simplex. We then show how this kind of tree can be used to efficiently evaluate $\alpha$-vectors over sets of belief points (or belief regions).

## 4 Metric-trees for belief spaces

Metric data structures offer a way to organize large sets of data points according to distances between the points. By organizing the data appropriately, it is possible to satisfy many different statistical queries over the elements of the set, without explicitly considering all points. Instances of metric data structures such as KD-trees, ball-trees and metric-trees have been shown to be useful for a wide range of learning tasks (e.g. nearest-neighbor, kernel regression, mixture modeling), including some with high-dimensional and non-Euclidean spaces. The metric-tree [12] in particular offers a very general approach to the problem of structural data partitioning. It consists of a hierarchical tree built by recursively splitting the set of points into spatially tighter subsets, assuming only that the distance between points is a metric.

### 4.1 Building a metric-tree from belief points

Each node $\eta$ in a metric-tree is represented by its center $\eta_c$, its radius $\eta_r$, and a set of points $\eta_B$ that fall within its radius. To recursively construct the tree—starting with node $\eta$ and building children nodes $\eta^1$ and $\eta^2$—we first pick two candidate centers (one per child) at the extremes of the $\eta$'s region: $\eta_c^1 = \max_{b \in \eta_D} D(\eta_c, b)$, and $\eta_c^2 = \max_{b \in \eta_D} D(\eta_c^1, b)$. In a single-step approximation to k-nearest-neighbor (k=2), we then re-allocate each point in $\eta_B$ to the child with the closest center (ties are broken randomly):

$$\eta_B^1 \leftarrow b \qquad \text{if } D(\eta_c^1, b) < D(\eta_c^2, b) \tag{5}$$
$$\eta_B^2 \leftarrow b \qquad \text{if } D(\eta_c^1, b) > D(\eta_c^2, b)$$

Finally we update the centers and calculate the radius for each child:

$$\eta_c^1 = \text{Center}\{\eta_B^1\} \qquad \eta_c^2 = \text{Center}\{\eta_B^2\} \tag{6}$$
$$\eta_r^1 = \max_{b \in \eta_B^1} D(\eta_c^1, b) \qquad \eta_r^2 = \max_{b \in \eta_B^2} D(\eta_c^2, b) \tag{7}$$

The general metric-tree algorithm allows a variety of ways to calculate centers and distances. For the centers, the most common choice is the centroid of the points and this is what we use when building a tree over belief points. We have tried other options, but with negligible impact. For the distance metric, we select the max-norm: $D(\eta_c, b) = ||\eta_c - b||_\infty$, which allows for fast searching as described in the next section. While the radius determines the size of the region enclosed by each node, the choice of distance metric determines its shape (e.g. with Euclidean distance, we would get hyper-balls of radius $\eta_r$). In the case of the max-norm, each node defines an $|S|$-dimensional hyper-cube of length $2*\eta_r$. Figure 2 shows how the first two-levels of a tree are built, assuming a 3-state problem.

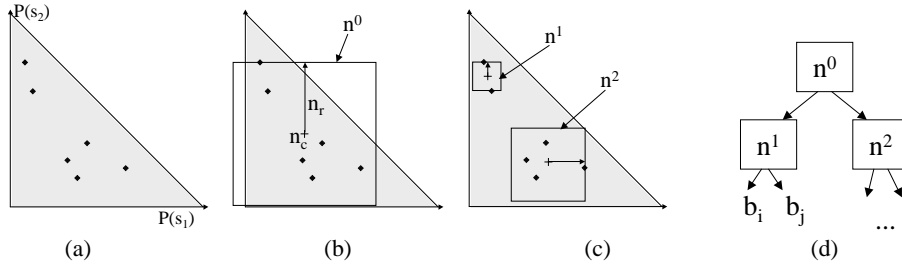

| (a) | (b) | (c) | (d) |

Figure 2: (a) Belief points. (b) Top node. (c) Level-1 left and right nodes. (d) Corresponding tree

While we need to compute the center and radius for each node to build the tree, there are additional statistics which we also store about each node. These are specific to using trees in the context of belief-state planning, and are necessary to evaluate $\alpha$ vectors over regions of the belief simplex. For a given node $\eta$ containing data points $\eta_B$, we compute $\eta_{min}$ and $\eta_{max}$, the vectors containing respectively the min and max belief in each dimension:

$$\eta_{min}(s) = \min_{b \in \eta_B} b(s), \forall s \in S \qquad \eta_{max}(s) = \max_{b \in \eta_B} b(s), \forall s \in S \qquad (8)$$

## 4.2 Searching over sub-regions of the simplex

Once the tree is built, it can be used for fast statistical queries. In our case, the goal is to compute $\text{argmax}_{\alpha \in \Gamma^{a,z}}(\alpha \cdot b)$ for all belief points. To do this, we consider the $\alpha$ vectors one at a time, and decide whether a new candidate $\alpha_i$ is better than any of the previous vectors $\{\alpha_0 \ldots \alpha_{i-1}\}$. With the belief points organized in a tree, we can often assess this over sets of points by consulting a high-level node $\eta$, rather than by assessing this for each belief point separately.

We start at the root node of the tree. There are four different situations we can encounter as we traverse the tree: first, there might be no single previous $\alpha$-vector that is best for all belief points below the current node (Fig. 3a). In this case we proceed to the children of the current node without performing any tests. In the other three cases there is a single dominant $alpha$-vector at the current node; the cases are that the newest vector $\alpha_i$ dominates it (Fig. 3b), is dominated by it (Fig. 3c), or neither (Fig. 3d). If we can prove that $\alpha_i$ dominates or is dominated by the previous one, we can prune the search and avoid checking the current node's children; otherwise we must check the children recursively.

We seek an efficient test to determine whether one vector, $\alpha_i$, dominates another, $\alpha_j$, over the belief points contained within a node. The test must be conservative: it must never erroneously say that one vector dominates another. It is acceptable for the test to miss some pruning opportunities—the consequence is an increase in run-time as we check more nodes than necessary—but this is best avoided if possible. The most thorough test would check whether $\Delta \cdot b$ is positive or negative at every belief sample $b$ under the current node

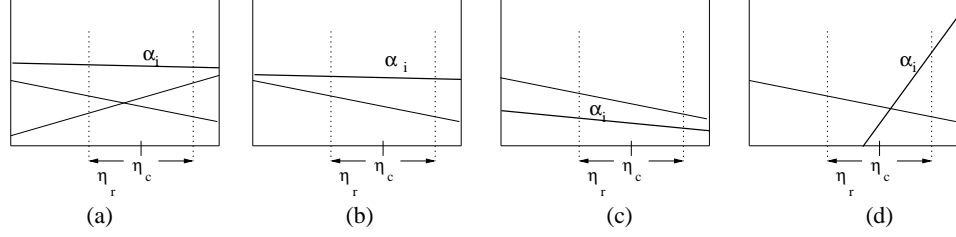

Figure 3: Possible scenarios when evaluation a new vector $\alpha$ at a node $\eta$, assuming a 2-state domain. (a) $\eta$ is a split node. (b) $\alpha_i$ is dominant. (c) $\alpha_i$ is dominated. (d) $\alpha_i$ is partially dominant.

(where $\Delta = (\alpha_i - \alpha_j)$). All positive would mean that $\alpha_i$ dominates $\alpha_j$, all negative the reverse, and mixed positive and negative would mean that neither dominates the other. Of course, this test renders the tree useless, since all points are checked individually. Instead, we test whether $\Delta \cdot b$ is positive or negative over a convex region $R$ which includes all of the belief samples that belong to the current node. The smaller the region, the more accurate our test will be; on the other hand, if the region is too complicated we won't be able to carry out the test efficiently. (Note that we can always test some region $R$ by solving one linear program to find $l = \min_{b \in R} b \cdot \Delta$, another to find $h = \max_{b \in R} b \cdot \Delta$, and testing whether $l < 0 < h$. But this is expensive and we prefer a more efficient test.)

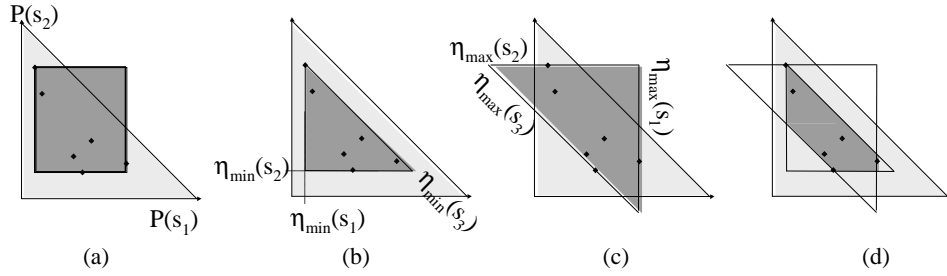

Figure 4: Several possible convex regions over subsets of belief points, assuming a 3-state domain.

We tested several types of region. The simplest type is an axis-parallel bounding box (Fig. 4a), $\eta_{\min} \leq b \leq \eta_{\max}$ for vectors $\eta_{\min}$ and $\eta_{\max}$ (as defined in Eq. 8). We also tested the simplex defined by $b \geq \eta_{\min}$ and $\sum_{s \in S} b(s) = 1$ (Fig. 4b), as well as the simplex defined by $b \leq \eta_{\max}$ and $\sum_{s \in S} b(s) = 1$ (Fig. 4c). The most effective test we discovered assumes $R$ is the intersection of the bounding box $\eta_{\min} \leq b \leq \eta_{\max}$ with the plane $\sum_{s \in S} b(s) = 1$ (Fig. 4d). For each of these shapes, minimizing or maximizing $b \cdot \Delta$ takes time $O(d)$ (where $d$=#states): for the box (Fig. 4a) we check each dimension independently, and for the simplices (Figs 4b, 4c) we check each corner exhaustively. For the last shape (Fig. 4d), maximizing with respect to $b$ is the same as computing $\delta$ s.t. $b(s) = \eta_{\min}(s)$ if $\Delta(s) < \delta$ and $b(s) = \eta_{\max}(s)$ if $\Delta(s) > \delta$. We can find $\delta$ in expected time $O(d)$ using a modification of the quick-median algorithm. In practice, not all $O(d)$ algorithms are equivalent. Empirical results show that checking the corners of regions (b) and (c) and taking the tightest bounds provides the fastest algorithm. This is what we used for the results presented below.

## 5 Results and Discussion

We have conducted a set of experiments to test the effectiveness of the tree structure in reducing computations. While still preliminary, these results illustrate a few interesting

properties of metric-trees when used in conjunction with point-based POMDP planning. Figure 5 presents results for six well-known POMDP problems, ranging in size from 4 to 870 states (for problem descriptions see [2], except for Coffee [10] and Tag [8]). While all these problems have been successfully solved by previous approaches, it is interesting to observe the level of speed-up that can be obtained by leveraging metric-tree data structures. In Fig. 5(a)-(f) we show the number of $B \times \Gamma$ (point-to-vector) comparisons required, with and without a tree, for different numbers of belief points. In Fig. 5(g)-(h) we show the computation time (as a function of the number of belief points) required for two of the problems. The *No-Tree* results were generated by applying the original PBVI algorithm (Section 2, [8]). The *Tree* results (which count comparisons on both internal and leaf nodes) were generated by embedding the tree searching procedure described in Section 4.2 within the same point-based POMDP algorithm. For some of the problems, we also show performance using an $\epsilon$-tree, where the test for vector dominance can reject (i.e. declare $\alpha_i$ is *dominated*, Fig. 3c) a new vector that is within $\epsilon$ of the current best vector.

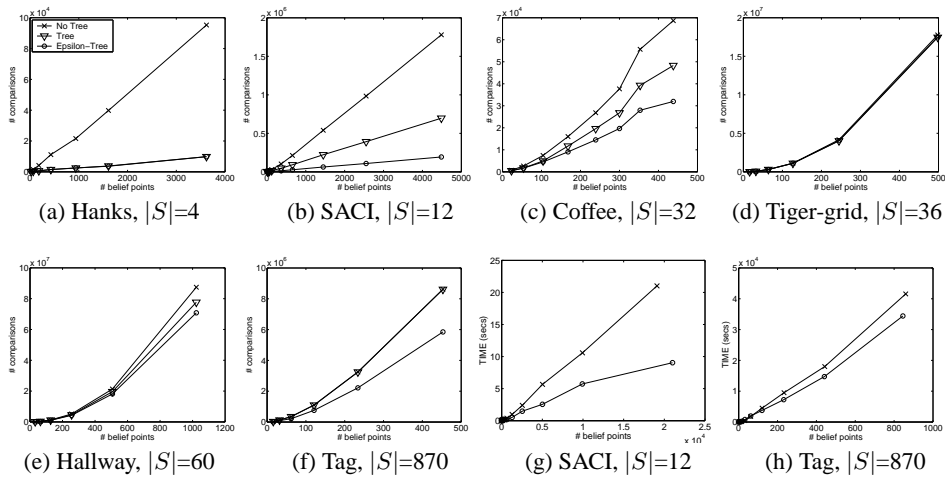

(a) Hanks, $|S|$=4    (b) SACI, $|S|$=12    (c) Coffee, $|S|$=32    (d) Tiger-grid, $|S|$=36

(e) Hallway, $|S|$=60    (f) Tag, $|S|$=870    (g) SACI, $|S|$=12    (h) Tag, $|S|$=870

Figure 5: Results of PBVI algorithm with and without metric-tree.

These early results show that, in various proportions, the tree can cut down on the number of comparisons. This illustrates how the use of metric-trees can effectively reduce POMDP computational load. The $\epsilon$-tree is particularly effective at reducing the number of comparisons in some domains (e.g. SACI, Tag). The much smaller effect shown in the other problems may be attributed to a poorly tuned $\epsilon$ (we used $\epsilon = 0.01$ in all experiments). The question of how to set $\epsilon$ such that we most reduce computation, while maintaining good control performance, tends to be highly problem-dependent.

In keeping with other metric-tree applications, our results show that computational savings increase with the number of belief points. What is more surprising is to see the trees paying off with so few data points (most applications of KD-trees start seeing benefits with 1000+ data points.) This may be partially attributed to the compactness of our convex test region (Fig. 4d), and to the fact that we do not search on split nodes (Fig. 3a); however, it is most likely due to the nature of our search problem: many $\alpha$ vectors are accepted/rejected before visiting *any* leaf nodes, which is different from typical metric-tree applications. We are particularly encouraged to see trees having a noticeable effect with very few data points because, in some domains, good control policies can also be extracted with few data points.

We notice that the effect of using trees is negligible in some larger problems (e.g. Tiger-grid), while still pronounced in others of equal or larger size (e.g. Coffee, Tag). This is

likely due to the intrinsic dimensionality of each problem.[3] Metric-trees often perform well in high-dimensional datasets with low intrinsic dimensionality; this also appears to be true of metric-trees applied to vector sorting. While this suggests that our current algorithm is not as effective in problems with intrinsic high-dimensionality, a slightly different tree structure or search procedure may well help in those cases. Recent work has proposed new kinds of metric-trees that can better handle point-based searches in high-dimensions [7], and some of this may be applicable to the POMDP $\alpha$-vector sorting problem.

## 6 Conclusion

We have described a new type of metric-tree which can be used for sorting belief points and accelerating value updates in POMDPs. Early experiments indicate that the tree structure, by appropriately pruning unnecessary $\alpha$-vectors over large regions of the belief, can accelerate planning for a range problems. The promising performance of the approach on the Tag domain opens the door to larger experiments.

### Acknowledgments

This research was supported by DARPA (MARS program) and NSF (ITR initiative).

## Footnotes

[1] In exact updates, this step requires taking a cross-sum over observations, which is $O(|S||A||V'|^{|Z|})$. By operating over a finite set of points, the cross-sum reduces to a simple sum, which is the main reason behind the computational speed-up obtained in point-based algorithms.

[2]Step 1 projects all vectors $\alpha \in V'$ for any $(a, z)$ pair. In the worse-case, this has time-complexity $O(|A| |Z| |S|^2 |V'|)$, however most problems have very sparse transition matrices and this is typically much closer to $O(|A| |Z| |S| |V'|)$. Step 3 is also relatively efficient at $O(|A| |Z| |S| |B|)$.

[3]The coffee domain is known to have an intrinsic dimensionality of 7 [10]. We do not know the intrinsic dimensionality of the Tag domain, but many robot applications produce belief points that exist in sub-dimensional manifolds [11].

## References

[1] R. I. Brafman. A heuristic variable grid solution method for POMDPs. In *Proceedings of the Fourteenth National Conference on Artificial Intelligence (AAAI)*, pages 727–733, 1997.

[2] A. Cassandra. http://www.cs.brown.edu/research/ai/pomdp/examples/index.html.

[3] J. H. Friendman, J. L. Bengley, and R. A. Finkel. An algorithm for finding best matches in logarithmic expected time. *ACM Transactions on Mathematical Software*, 3(3):209–226, 1977.

[4] M. Hauskrecht. Value-function approximations for partially observable Markov decision processes. *Journal of Artificial Intelligence Research*, 13:33–94, 2000.

[5] L. P. Kaelbling, M. L. Littman, and A. R. Cassandra. Planning and acting in partially observable stochastic domains. *Artificial Intelligence*, 101:99–134, 1998.

[6] A. W. Moore. Very fast EM-based mixture model clustering using multiresolution KD-trees. In *Advances in Neural Information Processing Systems (NIPS)*, volume 11, 1999.

[7] A. W. Moore. The anchors hierarchy: Using the triangle inequality to survive high dimensional data. Technical Report CMU-RI-TR-00-05, Carnegie Mellon, 2000.

[8] J. Pineau, G. Gordon, and S. Thrun. Point-based value iteration: An anytime algorithm for POMDPs. In *International Joint Conference on Artificial Intelligence (IJCAI)*, 2003.

[9] K.-M. Poon. A fast heuristic algorithm for decision-theoretic planning. Master's thesis, The Hong-Kong University of Science and Technology, 2001.

[10] P. Poupart and C. Boutilier. Value-directed compression of POMDPs. In *Advances in Neural Information Processing Systems (NIPS)*, volume 15, 2003.

[11] N. Roy and G. Gordon. Exponential family PCA for belief compression in POMDPs. In *Advances in Neural Information Processing Systems (NIPS)*, volume 15, 2003.

[12] J. K. Uhlmann. Satisfying general proximity/similarity queries with metric trees. *Information Processing Letters*, 40:175–179, 1991.

[13] R. Zhou and E. A. Hansen. An improved grid-based approximation algorithm for POMDPs. In *Proceedings of the 17th International Joint Conference on Artificial Intelligence (IJCAI)*, 2001.

